# The Neurodynamics of Belief Propagation on Binary Markov Random Fields

**Thomas Ott**
Institute of Neuroinformatics
ETH/UNIZH Zurich
Switzerland
tott@ini.phys.ethz.ch

**Ruedi Stoop**
Institute of Neuroinformatics
ETH/UNIZH Zurich
Switzerland
ruedi@ini.phys.ethz.ch

## Abstract

We rigorously establish a close relationship between message passing algorithms and models of neurodynamics by showing that the equations of a continuous Hopfield network can be derived from the equations of belief propagation on a binary Markov random field. As Hopfield networks are equipped with a Lyapunov function, convergence is guaranteed. As a consequence, in the limit of many weak connections per neuron, Hopfield networks exactly implement a continuous-time variant of belief propagation starting from message initialisations that prevent from running into convergence problems. Our results lead to a better understanding of the role of message passing algorithms in real biological neural networks.

## 1 Introduction

Real brain structures employ inference algorithms as a basis of decision making. Belief Propagation (BeP) is a popular, widely applicable inference algorithm that seems particularly suited for a neural implementation. The algorithm is based on message passing between distributed elements that resembles the signal transduction within a neural network. The analogy between BeP and neural networks is emphasised if BeP is formulated within the framework of Markov random fields (MRF). MRF are related to spin models [1] that are often used as abstract models of neural networks with symmetric synaptic weights. If a neural implementation of BeP can be realised on the basis of MRF, each neuron corresponds to a message passing element (hidden node of a MRF) and the synaptic weights reflect their pairwise dependencies. The neural activity then would encode the messages that are passed between connected nodes. Due to the highly recurrent nature of biological neural networks, MRF obtained in this correspondence to a neural network are naturally very "loopy". Convergence of BeP on loopy structures is, however, a delicate matter [1]-[2] .

Here, we show that BeP on binary MRF can be reformulated as continuous Hopfield networks along the lines of the sketched correspondence. More precisely, the equations of a continuous Hopfield network are derived from the equations of BeP on a binary MRF, if there are many, but weak connections per neuron. As a central result in this case, attractive fixed points of the Hopfield network provide very good approximations of BeP fixed points of the corresponding MRF. In the Hopfield case a Lyapunov function guarantees the convergence towards these fixed points. As a consequence, Hopfield networks implement BeP with guaranteed convergence. The result of the inference is directly represented by the activity of the neurons in the steady state. To illustrate this mechanism, we compare the magnetisations obtained in the original BeP framework to that from the Hopfield network framework, for a symmetric ferromagnetic model.

Hopfield networks may also serve as a guideline for the implementation or the detection of BeP in more realistic, e.g., spiking, neural networks. By giving up the symmetric synaptic weights constraints, we may generalise the original BeP inference algorithm towards capturing neurally inspired message passing.

## 2    A Quick Review on Belief Propagation in Markov Random Fields

MRF have been used to formulate inference problems, e.g. in Boltzmann machines (which actually are MRF [3]) or in the field of computer vision [4] and are related to Bayesian networks. In fact, both concepts are equivalent variants of graphical models [1]. Typically, from a given set of observations $\{y\}$, we want to infer some hidden quantities $\{x\}$ that, in our case, take on either of the two values $\{-1, 1\}$. For instance, the pixel values of a grey-scaled image may be represented by $\{y\}$, whereas a particular variable $x_i$ describes whether pixel $i$ belongs to an object ($x_i = 1$) or to the background ($x_i = -1$). The natural question that emerges in this context is: *Given the observations $\{y_i\}$, what is the probability for $x_i = 1$?* The relation between $\{y\}$ and $\{x\}$ is usually given by a joint probability, written in the factorised form

$$p(\{x\}, \{y\}) = p(\{x\}) = \frac{1}{Z} \prod_{(i,j)} \psi_{ij}(x_i, x_j) \prod_i \phi_i(x_i, y_i), \qquad (1)$$

where the functions $\{\psi\}$ describe the pairwise dependencies of the hidden variables $\{x\}$ and the functions $\{\phi\}$ give the evidences from $\{y\}$. $Z$ is the normalisation constant [1]. (1) can directly be reformulated as an Ising system with the Energy

$$E(s) = -\sum_{(i,j)} J_{ij}(s_i s_j) - \sum_i h_i(s_i), \qquad (2)$$

where the Boltzmann distribution provides the probability $p(s)$ of a spin configuration $s$,

$$p(s) = \frac{1}{Z} e^{-E(s)/T}. \qquad (3)$$

A comparison with (1) yields $s_i = x_i$, $J_{ij}(s_i, s_j)/T = \ln \psi_{ij}(x_i, x_j)$ and $h_i(s_i)/T = \ln \phi(x_i, y_i)$. In many cases, it is reasonable to assume that $J_{ij}(s_i, s_j) = J_{ij} s_i s_j = J_{ji} s_i s_j$ and that $h_i(s_i) = h_i s_i$, where $J_{ij}$ and $h_i$ are real-valued constants, so that (2) transforms into the familiar Ising Hamiltonian [5]. For convenience, we set $T = 1$.

The inference task inherent to MRF amounts to extracting marginal probabilities

$$p_i(x_i) = \sum_{x_k, k \neq i} p(\{x\}). \qquad (4)$$

An exact evaluation of $p_i$ according to Eq. (4) is generally very time-consuming. BeP provides us with approximated marginals within a reasonable time. This approach is based on the idea that connected elements (where a connection is given by $J_{ij} \neq 0$) interchange messages that contain a recommendation about what state the other elements should be in [1]. Given the set of messages $\{m_{ij}^t(x_j)\}$ at time $t$, the messages at time $t + 1$ are determined by

$$m_{ij}^{t+1}(x_j) = \sum_{x_i} \phi_i(x_i, y_i) \psi_{ij}(x_i, x_j) \prod_{k \in N(i) \backslash j} m_{ki}^t(x_i). \qquad (5)$$

Here, $m_{ij}$ denotes the message sent from the hidden variable (or node) $i$ to node $j$. $N(i) \backslash j$ denotes the set of all neighbouring nodes of $i$ without $j$. Usually, the messages are normalised at every time step, i.e., $m_{ij}^t(1) + m_{ij}^t(-1) = 1$. After (5) has converged, the marginals $p_i$ are approximated by the so called beliefs $b_i$ that are calculated according to

$$b_i(x_i) = k \phi_i(x_i, y_i) \prod_{j \in N(i)} m_{ji}(x_i), \qquad (6)$$

where $k$ is a normalisation constant. In particular in connection with Ising systems, one is primarily interested in the quantity $m_i = b_i(1) - b_i(-1)$, the so-called local magnetisation.

For a detailed introduction of BeP on MRF we refer to [1].

## 3    BeP and the Neurodynamics of Hopfield Networks

The goal of this section is to establish the relationship between the update rule (5) and the dynamical equation of a continuous Hopfield network,

$$\frac{dv_i(t)}{dt} = -v_i(t) + f\left(\sum_k w_{ki} v_k(t)\right) + K_i(t). \qquad (7)$$

Here $v_i$ is some quantity describing the activity of neuron $i$ (e.g., the membrane potential) and $f(x)$ is the activation function, typically implemented in a sigmoid form, such as $f(x) = \tanh(x)$. $w_{ij} = w_{ji}$ are the connection (synaptic) weights which need to be symmetric in the Hopfield model. The connectivity might be all-to-all or sparse. $K_i(t)$ is an external signal or bias (see, e.g., [6] for a general introduction to Hopfield networks). According to the sketched picture, each neuron represents a node $x_i$, whereas the messages are encoded in the variables $v_i$ and $w_{ij}$. The exact nature of this encoding will be worked out below. The Hopfield architecture implements the point attractor paradigm, i.e., by means of the dynamics the network is driven into a fixed point. At the fixed point, the beliefs $b_i$ can be read out. In the MRF picture, this corresponds to (5) and (6). We will now realise the translation from MRF into Hopfield networks as follows:

(1) Reduction of the number of messages per connection from $m_{ij}(1)$ and $m_{ij}(-1)$ to one reparameterised variable $n_{ij}$.
(2) Translation into a continuous system.
(3) Translation of the obtained equations into the equations of a Hopfield network, where we find the encoding of the variables $n_{ij}$ in terms of $v_i$ and $w_{ij}$.

This will establish the exact relationship between Hopfield and BeP.

## 3.1 Reparametrisation of the messages

In the case of binary variables $x_i$, the messages $m_{ji}(x_i)$ can be reparameterised [2] according to

$$\tanh n_{ij} = m_{ij}(x_j = 1) - m_{ij}(x_j = -1). \tag{8}$$

By this, the update rules (5) transform into update rules for the new "messages" $n_{ij}$

$$f(\mathbf{n}) : n_{ij}^{t+1} = \tanh^{-1}\left[\tanh(J_{ij}) \tanh\left(\sum_{k \in N(i) \backslash j} n_{ki}^t + h_i\right)\right]. \tag{9}$$

For each connection $i \to j$ we obtain one single message $n_{ij}$. We can now directly calculate the local magnetisation according to $m_i = \tanh(\sum_{k \in N_i} n_{ki} + h_i)$ [2]. The Jacobian of (9) in a point $\mathbf{n}$ is denoted by $df(\mathbf{n}) = (\frac{\partial n_{ij}^{t+1}}{\partial n_{kl}^t})|_{\mathbf{n}}$.

The used reparametrisation translates the update rules into an additive form ("log domain") which is a basic assumption of most models of neural networks.

## 3.2 Translation into a time-continuous system

Eq. (9) can be translated into the equivalent time-continuous system

$$\frac{dn_{ij}(t)}{dt} = g_{ij}(\mathbf{n}(t)) = -n_{ij}(t) + \tanh^{-1}\left[\tanh(J_{ij}) \tanh\left(\sum_{k \in N(i) \backslash j} n_{ki}(t) + h_i(t)\right)\right], \tag{10}$$

where $h_i(t) = h_i$ is time-independent. The corresponding Jacobian in a point $\mathbf{n}$ is denoted by $dg(\mathbf{n}) = -Id + df(\mathbf{n})$, where $Id$ is the $|n|$-dimensional identity matrix ($|n|$ is the number of messages $n_{ij}$). Obviously, (9) and (10) have the same fixed points $\mathbf{n_{fp}}$ which are given by

$$n_{ij} = \tanh^{-1}\left[\tanh(J_{ij}) \tanh\left(\sum_{k \in N(i) \backslash j} n_{ki} + h_i\right)\right], \tag{11}$$

with identical stability properties in both frameworks: For stability of (9) it is required that the real part of the largest eigenvalue of the Jacobian $df(\mathbf{n_{fp}})$ be smaller than 1, whereas for the stability of (10) the condition is that the real part of the largest eigenvalue of $dg(\mathbf{n_{fp}}) = -Id + df(\mathbf{n_{fp}})$ must be smaller than 0. It is obvious that both conditions are identically satisfied.

### 3.3 Translation into a Hopfield network

The comparison between Eq. (7) and Eq. (10) does not lead to a direct identification of $v_i$ with $n_{ij}$. Rather, under certain conditions, we can identify $n_{ij}$ with $w_{ij}v_i$. That is, a message corresponds to the presynaptic neural activity weighted by the synaptic strength. Formally, we may define a variable $v_i^j$ by $n_{ij} = w_{ij}v_i^j$ and rewrite Eq. (10) as

$$\frac{d}{dt}w_{ij}v_i^j = -w_{ij}v_i^j + \tanh^{-1}\left[w_{ij}\tanh\left(\sum_{k \in N(i)} w_{ki}v_k^i - w_{ji}v_j^i + h_i(t)\right)\right], \qquad (12)$$

where we set $w_{ij} = \tanh(J_{ij})$.[1] In the following, we assume that the synaptic weights $w_{ij}$ are relatively small, i.e., $w_{ij} \ll 1$. Hence $\tanh^{-1}(x)$ can be approximated by $\tanh^{-1}(x) \approx x$. Moreover, if a neuron receives many inputs (number of connections $q_i \gg 1$) then the single contribution $w_{ji}v_j^i$ can be neglected. Thus (12) simplifies to

$$\frac{d}{dt}w_{ij}v_i^j = -w_{ij}v_i^j + w_{ij}\tanh\left(\sum_{k \in N(i)} w_{ki}v_k^i + h_i(t)\right). \qquad (13)$$

Upon a division by $w_{ij}$, we arrive at the equation

$$\frac{d}{dt}v_i^j = -v_i^j + \tanh\left(\sum_{k \in N(i)} w_{ki}v_k^i + h_i(t)\right) \qquad (14)$$

which for a uniform initialisation $v_i^1(0) = v_i^2(0) = ... = v_i^{q_i}(0)$ for all $i$ preserves this uniformity through time, i.e., $v_i^1(t) = v_i^2(t) = ... = v_i^{q_i}(t)$. In other words, the subset defined by $v_i^1 = v_i^2 = ...v_i^{q_i}$ is invariant under the dynamics of (14). For such an initialisation we can therefore replace for a $i$ all $v_i^j$ by a single variable $v_i$, which leads to the equation

$$\frac{dv_i}{dt} = -v_i + \tanh\left(\sum_{k \in N(i)} w_{ki}v_k + h_i(t)\right). \qquad (15)$$

Using $\tanh(x + y) \approx \tanh(x) + \tanh(y)$ if $y \ll 1$, and with $y = h_i$ we end up with the postulated equation (7). After the convergence to an attractor fixed point, the local magnetisation is simply the activity $v_i$. This is because the fixed point and the read out equations collapse under the approximation $\tanh(\sum_k w_{ki}v_k + h_i) \approx \tanh(\sum_k w_{ki}v_k) + K_i$, i.e., $v_i(t = \infty) = m_i$.

In summary, we can emulate the original BeP procedure by a continuous Hopfield network provided that (I) the single weights $w_{ij}$ and the external fields $h_i(t)$ are relatively weak, (II) that each neuron receives many inputs and (III) that the original messages have been initialised according to $v_i^{k_1}(0) = n_{ik_1}/w_{ik_1} = v_i^{k_2}(0) = n_{ik_2}/w_{ik_2} = ... = v_i^{k_{q_i}}(0) = n_{ik_{q_i}}/w_{ik_{q_i}}$. From a biological point of view, the first two points seem reasonable. The effect of a single synapse is typically small compared to the totality of the numerous synaptic inputs of a cell [7]-[8]. In this sense, single weights are considered weak. In order to establish a firm biological correspondence, particular consideration will be required for the last point. In the next section, we show that Hopfield networks are guaranteed to converge and thus, the required initialisation can be considered a natural choice for BeP on MRF with the properties (I) and (II).

### 3.4 Guarantee of convergence

A basic Hopfield model of the form

$$\frac{dx_i(t)}{dt} = -x_i(t) + \sum_j w_{ji}f(x_j(t)) + I_j, \qquad (16)$$

with $f(x) = \tanh(x)$, has the same attractor structure as the model (7) described above (see [6] and references therein). For the former model, an explicit Lyapunov function has been constructed

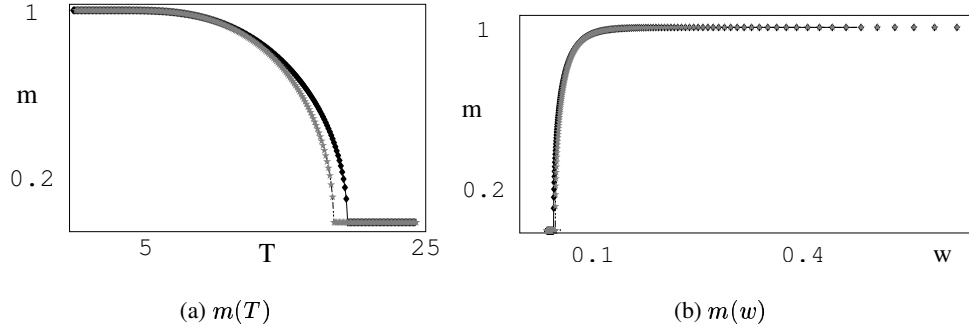

(a) $m(T)$                                          (b) $m(w)$

Figure 1: The magnetisation $m$ as a function of $T$ and $w$ for the symmetric ferromagnetic model. The results for the original BeP (grey stars) and for the Hopfield network (black circles) are compared.

[9] which assures that these networks and with them the networks considered by us are globally asymptotically stable [6].

Moreover, the time-continuous model (7) can be translated back into a time-discrete model, yielding

$$v_i(t+1) = \tanh\left(\sum_j w_{ji} v_j(t)\right) + K_i(t). \tag{17}$$

This equation is the proper analogue of Eq. (9).

## 4   Results for the Ferromagnetic Model

In this section, we evaluate the Hopfield-based inference solution $m_i = v_i(t = \infty)$ for networks with a simple connectivity structure: We assume constant positive synaptic weights $w = w_{ij}$ (ferromagnetic couplings) and a constant number of connections per neuron $q$. We furthermore abstain from an external field and set $K_i = 0$. To realise this symmetric model, we may either think of an infinitely extended network or of a network with some spatial periodicity, e.g., a network on a torus. According to the last section, $w$ is related to $J$ in a spin model via $w = \tanh(J) = \tanh(1/T)$, where, for convenience, we reintroduced a quasi-temperature $T$ as a scaling parameter.

From Eq. (7), it is clear that $\mathbf{v_{fp}} = (v_{fp}, v_{fp}, ..., v_{fp})$ is a fixed point of the system if $v_{fp} = \tanh(qwv_{fp})$. This equation has always a solution $v_{fp} = v_0 = 0$. However, the stability of $v_0$ is restricted to $T > T_{crit}^{hn}$, where the bifurcation point is given by

$$T_{crit}^{hn} = \frac{1}{\tanh^{-1}\left(\frac{1}{q}\right)}. \tag{18}$$

This follows from the critical condition $\frac{\partial \tanh(qwv)}{\partial v}\big|_{v=v_0} = 1$. For $T < T_{crit}^{hn}$, two additional and stable fixed points $v_\pm$ emerge which are symmetric with respect to the origin. After the convergence to a stable fixed point, $v_+$ for $T < T_{crit}^{hn}$ and $v_0$ for $T > T_{crit}^{hn}$, the obtained magnetisation $m = \tanh(qwv_{fp})$ equal to $v_{fp}$ is shown in dependence of $T$ in Fig. 1a (black circles), for $q = 20$. The critical point is found at a temperature $T_{crit}^{hn} = 1/\tanh^{-1}(1/20) = 19.98$

The result is compared to the result obtained on the basis of the original BeP equations (5) (grey stars in Fig 1a). We see that the critical point is slightly lower in the original BeP case. This can be understood from Eq. (9), for which the point given by the messages $\mathbf{n_0} = (0, 0, 0...)$ looses stability at the critical temperature

$$T_{crit}^{bep} = \frac{1}{\tanh^{-1}\left(\frac{1}{q-1}\right)}. \tag{19}$$

For the value $q = 20$, this yields $T_{crit}^{bep} = 18.98$. $T_{crit}^{bep}$ is in fact the critical temperature for Ising grids obtained in the Bethe-Peierls approximation (for $q = 4$, we get $T_{crit}^{bep} = 2.88539$ [5]). In

this way, we casually come across the deep relationship of BeP and Bethe-Peierls which has been established by the theorem stating that stable BeP fixed points are local minima of the Bethe free energy functional [1],[10].

In the limit of small weights, i.e. large $T$, the results for Hopfield nets and BeP must be identical. This, in fact, is certainly true for $T > T_{crit}^{hn}$, where $m = 0$ in both cases. For very large weights, i.e., small $T$, the results are also identical in the case of the ferromagnetic couplings studied here, as $m \to 1$. It is only around the critical values, where the two results seem to differ. A comparison of the results against the synaptic weight $w$, however, shows an almost perfect agreement for all $w$. The differences can be made arbitrarily small for larger $q$.

## 5  Discussion and Outlook

In this report, we outlined the general structural affinity between belief propagation on binary Markov random fields and continuous Hopfield networks. According to this analogy, synaptic weights correspond to the pairwise dependencies in the MRF and the neuronal signal transduction corresponds to the message exchange. In the limit of many synaptic connections per neuron, but comparatively small individual synaptic weights, the dynamics of the Hopfield network is an exact mirror of the BeP dynamics in its time-continuous form. To achieve the agreement, the choice of initial messages needs to be confined. From this we can conclude that Hopfield network attractors are also BeP attractors (whereas the opposite does not necessarily hold). Unlike BeP, Hopfield networks are guaranteed to converge to a fixed point. We may thus argue that Hopfield networks naturally implement useful message initialisations that prevent trapping into a limit cycle. As a further benefit, the local magnetisations, as the result of the inference process, are just reflected in the asymptotic neural activity. The binary basis of the implementation is not necessarily a drawback, but could simply reflect the fact that many decisions have a yes-or-no character.

Our work so far has preliminary character. The Hopfield network model is still a crude simplification of biological neural networks and the relevance of our results for such real-world structures remains somewhat open. However, the search for a possible neural implementation of BeP is appealing and different concepts have already been outlined [11]. This approach shares our guiding idea that the neural activity should directly be interpreted as a message passing process. Whereas our approach is a mathematically rigorous intermediate step towards more realistic models, the approach chosen in [11] tries to directly implement BeP with spiking neurons. In accordance with the guiding idea, our future work will comprise three major steps. First, we take the step from Hopfield networks to networks with spiking elements. Here, the question is to what extent can the concepts of message passing be adapted or reinterpreted so that a BeP implementation is possible. Second, we will give up the artificial requirement of symmetric synaptic weights. To do this, we might have to modify the original BeP concept, while we still may want to stick to the message passing idea. After all, there is no obvious reason why the brain should implement exactly the BeP algorithm. It rather seems plausible that the brain employs inference algorithms that might be conceptually close to BeP. Third, the context and the tasks for which such algorithms can actually be used must be elaborated. Furthermore, we need to explore how the underlying structure could actually be learnt by a neural system.

Message passing-based inference algorithms offer an attractive alternative to traditional notions of computation inspired by computer science, paving the way towards a more profound understanding of natural computation [12]. To judge its eligibility, there is - ultimately - one question: How can the usefulness (or inappropriateness) of the message passing concept in connection with biological networks be verified or challenged experimentally?

**Acknowledgements**

This research has been supported by a ZNZ grant (Neuroscience Center Zurich).

**References**

[1] Yedidia, J.S., Freeman, W.T., Weiss, Y. (2003) Understanding belief propagtion and its generalizations. In G. Lakemeyer and B. Nebel (eds.) *Exploring Artificial Intelligence in the New Millenium*, Morgan Kaufmann, San Francisco.

[2] Mooij, J.M., Kappen, H.J. (2005) On the properties of the Bethe approximation and loopy belief propagation on binary networks. *J.Stat.Mech.*, doi:10.1088/1742-5468/2005/11/P11012.

[3] Welling, M., Teh, W.T. (2003) Approximate inference in Boltzmann machines. *Artificial Intelligence* **143**:19-50.

[4] Geman, S., Geman, D. (1984) Stochastic relaxation, Gibbs distributions, and the Bayesian restoration of images. *IEEE-PAMI* **6**(6):721-741.

[5] Huang, K. (1987) Statistical mechanics. Second edition, John Wiley & Sons, New York, Chapter 13.

[6] Haykin, S. (1999) Neural networks - a comprehensive foundation. Second edition, Prentice-Hall, Inc., Chapter 14.

[7] Koch, C. (1999), Biophysics of computation. Oxford University Press, Inc., New York.

[8] Douglas, R.J., Mahowald, M., Martin, K.A.C., Stratford, K.J. (1996) The role of synapses in cortical computation. *Journal of Neurocytology* **25**: 893-911.

[9] Hopfield, J.J. (1984) Neurons with graded response have collective computational properties like those of two-state neurons. *PNAS* **81**:3088-3092.

[10] Heskes, T. (2004) On the uniqueness of loopy belief propagation fixed points. *Neural Comput.* **16**:2379-2413.

[11] Shon, A.P., Rao, R.P.N. (2005) Implementing belief propagation in neural circuits. *Neurocomputing* **65-66**:877-884.

[12] Stoop, R., Stoop, N. (2004) Natural computation measured as a reduction of complexity. *Chaos* **14**(3):675-679.

## Footnotes

[1]Hence the synaptic weights $w_{ij}$ are automatically restricted to the interval $]-1, 1[$.
